# Harmonic Grammars
# for Formal Languages

**Paul Smolensky**
Department of Computer Science &
Institute of Cognitive Science
University of Colorado
Boulder, Colorado 80309-0430

## Abstract

Basic connectionist principles imply that grammars should take the form of systems of parallel soft constraints defining an optimization problem the solutions to which are the well-formed structures in the language. Such *Harmonic Grammars* have been successfully applied to a number of problems in the theory of natural languages. Here it is shown that formal languages too can be specified by Harmonic Grammars, rather than by conventional serial re-write rule systems.

## 1 HARMONIC GRAMMARS

In collaboration with Géraldine Legendre, Yoshiro Miyata, and Alan Prince, I have been studying how symbolic computation in human cognition can arise naturally as a higher-level virtual machine realized in appropriately designed lower-level connectionist networks. The basic computational principles of the approach are these:

(1)    a. When analyzed at the lower level, mental representations are distributed patterns of connectionist activity; when analyzed at a higher level, these same representations constitute symbolic structures. The particular symbolic structure **s** is characterized as a set of *filler/role bindings* $\{f_i/r_i\}$, using a collection of structural roles $\{r_i\}$ each of which may be occupied by a filler $f_i$—a constituent symbolic struc-

ture. The corresponding lower-level description is an activity vector $\mathbf{s} = \sum_i \mathbf{f}_i \otimes \mathbf{r}_i$. These *tensor product representations* can be defined recursively: fillers which are themselves complex structures are represented by vectors which in turn are recursively defined as tensor product representations. (Smolensky, 1987; Smolensky, 1990).

b. When analyzed at the lower level, mental processes are massively parallel numerical activation spreading; when analyzed at a higher level, these same processes constitute a form of symbol manipulation in which entire structures, possibly involving recursive embedding, are manipulated in parallel. (Dolan and Smolensky, 1989; Legendre et al., 1991a; Smolensky, 1990).

c. When the lower-level description of the activation spreading processes satisfies certain mathematical properties, this process can be analyzed on a higher level as the construction of that symbolic structure including the given input structure which *maximizes Harmony* (equivalently, minimizes 'energy'. The Harmony can be computed either at the lower level as a particular mathematical function of the numbers comprising the activation pattern, or at the higher level as a function of the symbolic constituents comprising the structure. In the simplest cases, the core of the Harmony function can be written at the lower, connectionist level simply as the quadratic form $H = \mathbf{a}^T \mathbf{W} \mathbf{a}$, where $\mathbf{a}$ is the network's activation vector and $\mathbf{W}$ its connection weight matrix. At the higher level, $H = \sum_{c_1, c_2} H_{c_1; c_2}$; each $H_{c_1; c_2}$ is the Harmony of having the two symbolic constituents $c_1$ and $c_2$ in the same structure (the $c_i$ are constituents *in particular structural roles*, and may be the same). (Cohen and Grossberg, 1983; Golden, 1986; Golden, 1988; Hinton and Sejnowski, 1983; Hinton and Sejnowski, 1986; Hopfield, 1982; Hopfield, 1984; Hopfield, 1987; Legendre et al., 1990a; Smolensky, 1983; Smolensky, 1986).

Once Harmony (connectionist well-formedness) is identified with grammaticality (linguistic well-formedness), the following results (1c) (Legendre et al., 1990a):

(2)  a. The explicit form of the Harmony function can be computed to be a sum of terms each of which measures the well-formedness arising from the coexistence, within a single structure, of a pair of constituents in their particular structural roles.

b. A (descriptive) grammar can thus be identified as a set of *soft rules* each of the form:
> If a linguistic structure $S$ simultaneously contains constituent $c_1$ in structural role $r_1$ and constituent $c_2$ in structural role $r_2$, then add to $H(S)$, the Harmony value of $S$, the quantity $H_{c_1, r_1; c_2, r_2}$ (which may be positive or negative).

A set of such soft rules (or "constraints," or "preferences") defines a *Harmonic Grammar*.

c. The constituents in the soft rules include both those that are given in the input and the "hidden" constituents that are assigned to the input by the grammar. The problem for the parser (computational

grammar) is to construct that structure $S$, containing both input and "hidden" constituents, with the highest overall Harmony $H(S)$.

Harmonic Grammar (HG) is a formal development of conceptual ideas linking Harmony to linguistics which were first proposed in Lakoff's *cognitive phonology* (Lakoff, 1988; Lakoff, 1989) and Goldsmith's *harmonic phonology* (Goldsmith, 1990; Goldsmith, in press). For an application of HG to natural language syntax/semantics, see (Legendre et al., 1990a; Legendre et al., 1990b; Legendre et al., 1991b; Legendre et al., in press). Harmonic Grammar has more recently evolved into a non-numerical formalism called *Optimality Theory* which has been successfully applied to a range of problems in phonology (Prince and Smolensky, 1991; Prince and Smolensky, in preparation). For a comprehensive discussion of the overall research program see (Smolensky et al., 1992).

## 2   HGs FOR FORMAL LANGUAGES

One means for assessing the expressive power of Harmonic Grammar is to apply it to the specification of formal languages. Can, e.g., any Context-Free Language (CFL) $L$ be specified by an HG? Can a set of soft rules of the form (2b) be given so that a string $s \in L$ iff the maximum-Harmony tree with $s$ as terminals has, say, $H \geq 0$? A crucial limitation of these soft rules is that each may only refer to a *pair* of constituents: in this sense, they are only *second order*. (It simplifies the exposition to describe as "pairs" those in which both constituents are the same; these actually correspond to first order soft rules, which also exist in HG.)

For a CFL, a tree is well-formed iff all of its *local trees* are—where a local tree is just some node and all its children. Thus the HG rules need only refer to pairs of nodes which fall in a single local tree, i.e., parent-child pairs and/or sibling pairs. The $H$ value of the entire tree is just the sum of all the numbers for each such pair of nodes given by the soft rules defining the HG.

It is clear that for a general context-free grammar (CFG), pairwise evaluation doesn't suffice. Consider, e.g., the following CFG fragment, $G_0$ : $A{\rightarrow}B\ C$, $A{\rightarrow}D\ E$, $F{\rightarrow}B\ E$, and the ill-formed local tree $(A\ ;\ (B\ E))$ (here, $A$ is the parent, $B$ and $E$ the two children). Pairwise well-formedness checks fail to detect the ill-formedness, since the first rule says $B$ can be a left child of $A$, the second that $E$ can be a right child of $A$, and the third that $B$ can be a left sibling of $E$. The ill-formedness can be detected only by examining *all three* nodes simultaneously, and seeing that this triple is not licensed by any single rule.

One possible approach would be to extend HG to rules higher than second order, involving more than two constituents; this corresponds to $H$ functions of degree higher than 2. Such $H$ functions go beyond standard connectionist networks with pairwise connectivity, requiring networks defined over hypergraphs rather than ordinary graphs. There is a natural alternative, however, that requires no change at all in HG, but instead adopts a special kind of grammar for the CFL. The basic trick is a modification of an idea taken from Generalized Phrase Structure Grammar (Gazdar et al., 1985), a theory that adapts CFGs to the study of natural languages.

It is useful to introduce a new normal form for CFGs, *Harmonic Normal Form*

(HNF). In HNF, all rules of are three types: $A[i] \rightarrow B\ C$, $A \rightarrow a$, and $A \rightarrow A[i]$; and there is the further requirement that there can be only one branching rule with a given left hand side—the *unique branching condition*. Here we use lowercase letters to denote terminal symbols, and have two sorts of non-terminals: general symbols like $A$ and *subcategorized* symbols like $A[1], A[2], ..., A[i]$. To see that every CFL $L$ does indeed have an HNF grammar, it suffices to first take a CFG for $L$ in Chomsky Normal Form, and, for each (necessarily binary) branching rule $A \rightarrow B\ C$, (i) replace the symbol $A$ on the left hand side with $A[i]$, using a different value of $i$ for each branching rule with a given left hand side, and (ii) add the rule $A \rightarrow A[i]$.

Subcategorizing the general category $A$, which may have several legal branching expansions, into the specialized subcategories $A[i]$, each of which has only one legal branching expansion, makes it possible to determine the well-formedness of an entire tree simply by examining each parent/child pair separately: an entire tree is well-formed iff every parent/child pair is. The unique branching condition enables us to evaluate the Harmony of a tree simply by adding up a collection of numbers (specified by the soft rules of an IIG), one for each node and one for each link of the tree. Now, any CFL $L$ can be specified by a Harmonic Grammar. First, find an HNF grammar $G_{HNF}$ for $L$; from it, generate a set of soft rules defining a Harmonic Grammar $G_H$ via the correspondences:

| $G_{HNF}$ | $G_H$ |
|---|---|
| $a$ | $R_a$: If $a$ is at any node, add $-1$ to $H$ |
| $A$ | $R_A$: If $A$ is at any node, add $-2$ to $H$ |
| $A[i]$ | $R_{A[i]}$: If $A[i]$ is at any node, add $-3$ to $H$ |
| start symbol $S$ | $R_{root}$: If $S$ is at the root, add $+1$ to $H$ |
| $A \rightarrow \alpha$ ($\alpha = a$ or $A[i]$) | If $\alpha$ is a left child of $A$, add $+2$ to $H$ |
| $A[i] \rightarrow B\ C$ | If $B$ is a left child of $A[i]$, add $+2$ to $H$ |
| | If $C$ is a right child of $A[i]$, add $+2$ to $H$ |

The soft rules $R_a$, $R_A$, $R_{A[i]}$ and $R_{root}$ are first-order and evaluate tree nodes; the remaining second-order soft rules are *legal domination* rules evaluating tree links.

This IIG assigns $H = 0$ to any legal parse tree (with $S$ at the root), and $H < 0$ for any other tree; thus $s \in L$ iff the maximal-Harmony completion of $s$ to a tree has $H \geq 0$.

> *Proof.* We evaluate the Harmony of any tree by conceptually breaking up its nodes and links into pieces each of which contributes either $+1$ or $-1$ to $H$. In legal trees, there will be complete cancellation of the positive and negative contributions; illegal trees will have uncancelled $-1$s leading to a total $H < 0$.
>
> The decomposition of nodes and links proceeds as follows. Replace each (undirected) link in the tree with a pair of directed links, one pointing up to the parent, the other down to the child. If the link joins a legal parent/child pair, the corresponding legal domination rule will contribute $+2$ to $H$; break this $+2$ into two contributions of $+1$, one for each of the directed links. We similarly break up the non-terminal nodes into sub-nodes. A non-terminal node labelled

$A[i]$ has two children in legal trees, and we break such a node into three sub-nodes, one corresponding to each downward link to a child and one corresponding to the upward link to the parent of $A[i]$. According to soft rule $R_{A[i]}$, the contribution of this node $A[i]$ to $H$ is $-3$; this is distributed as three contributions of $-1$, one for each sub-node. Similarly, a non-terminal node labelled $A$ has only one child in a legal tree, so we break it into two sub-nodes, one for the downward link to the only child, one for the upward link to the parent of $A$. The contribution of $-2$ dictated by soft rule $R_A$ is similarly decomposed into two contributions of $-1$, one for each sub-node. There is no need to break up terminal nodes, which in legal trees have only one outgoing link, upward to the parent; the contribution from $R_a$ is already just $-1$.

We can evaluate the Harmony of any tree by examining each node, now decomposed into a set of sub-nodes, and determining the contribution to $H$ made by the node and its *outgoing* directed links. We will not double-count link contributions this way; half the contribution of each original undirected link is counted at each of the nodes it connects.

Consider first a non-terminal node $n$ labelled by $A[i]$; if it has a legal parent, it will have an upward link to the parent that contributes $+1$, which cancels the $-1$ contributed by $n$'s corresponding sub-node. If $n$ has a legal left child, the downward link to it will contribute $+1$, cancelling the $-1$ contributed by $n$'s corresponding sub-node. Similarly for the right child. Thus the total contribution of this node will be 0 if it has a legal parent and two legal children. For each *missing* legal child or parent, the node contributes an uncancelled $-1$, so the contribution of this node $n$ in the general case is:

(3)  $H_n = -$(the number of missing legal children and parents of node $n$)

The same result (3) holds of the non-branching non-terminals labelled $A$; the only difference is that now the only child that could be missing is a legal left child. If $A$ happens to be a legal start symbol in root position, then the $-1$ of the sub-node corresponding to the upward link to a parent is cancelled not by a legal parent, as usual, but rather by the $+1$ of the soft rule $R_{root}$. The result (3) still holds even in this case, if we simply agree to count the root position itself as a legal parent for start symbols. And finally, (3) holds of a terminal node $n$ labelled $a$; such a node can have no missing child, but might have a missing legal parent.

Thus the total Harmony of a tree is $H = \sum_n H_n$, with $H_n$ given by (3). That is, $H$ is the *minus* the total number of missing legal children and parents for all nodes in the tree. Thus, $H = 0$ if each node has a legal parent and all its required legal children, otherwise $H \leq 0$. Because the grammar is in Harmonic Normal Form, a parse tree is legal iff every every node has a legal parent and its required

number of legal children, where "legal" parent/child dominations are defined only pairwise, in terms of the parent and one child, blind to any other children that might be present or absent. Thus we have established the desired result, that the maximum-Harmony parse of a string $s$ has $H \geq 0$ iff $s \in L$.

We can also now see how to understand the soft rules of $G_H$, and how to generalize beyond Context-Free Languages. The soft rules say that each node makes a negative contribution equal to its valence, while each link makes a positive contribution equal to its valence (2); where the "valence" of a node (or link) is just the number of links (or nodes) it is attached to in a legal tree. The negative contributions of the nodes are made any time the node is present; these are cancelled by positive contributions from the links only when the link constitutes a legal domination, sanctioned by the grammar.

So in order to apply the same strategy to unrestricted grammars, we will simply set the magnitude of the (negative) contributions of nodes equal to their valence, as determined by the grammar. □

We can illustrate the technique by showing how HNF solves the problem with the simple three-rule grammar fragment $G_0$ introduced early in this section. The corresponding HNF grammar fragment $G_{HNF}$ given by the above construction is $A[1] \rightarrow B\ C$, $A \rightarrow A[1]$, $A[2] \rightarrow D\ E$, $A \rightarrow A[2]$, $F[1] \rightarrow B\ E$, $F \rightarrow F[1]$. To avoid extraneous complications from adding a start node above and terminal nodes below, suppose that both $A$ and $F$ are valid start symbols and that $B$, $C$, $D$, $E$ are terminal nodes. Then the corresponding HG $G_H$ assigns to the ill-formed tree $(A\ ;\ (B\ E))$ the Harmony $-4$, since, according to $G_{HNF}$, $B$ and $E$ are both missing a legal parent and $A$ is missing two legal children. Introducing a now-necessary subcategorized version of $A$ helps, but not enough: $(A\ ;\ (A[1]\ ;\ (B\ E)))$ and $(A\ ;\ (A[2]\ ;\ (B\ E)))$ both have $H = -2$ since in each, one leaf node is missing a legal parent ($E$ and $B$, respectively), and the $A[i]$ node is missing the corresponding legal child. But the correct parse of the string $B\ E$, $(F\ ;\ (F[1]\ ;\ (B\ E)))$, has $H = 0$.

This technique can be generalized from context-free to unrestricted (type 0) formal languages, which are equivalent to Turing Machines in the languages they generate (e.g., (Hopcroft and Ullman, 1979)). The $i$th production rule in an unrestricted grammar, $R_i : \alpha_1\alpha_2 \cdots \alpha_{n_i} \rightarrow \beta_1\beta_2 \cdots \beta_{m_i}$ is replaced by the two rules: $R_i' : \alpha_1\alpha_2 \cdots \alpha_{n_i} \rightarrow \Gamma[i]$ and $R_i'' : \Gamma[i] \rightarrow \beta_1\beta_2 \cdots \beta_{m_i}$, introducing new non-terminal symbols $\Gamma[i]$. The corresponding soft rules in the Harmonic Grammar are then: "If the $k$th parent of $\Gamma[i]$ is $\alpha_k$, add $+2$ to $H$" and "If $\beta_k$ is the $k$th child of $\Gamma[i]$, add $+2$ to $H$"; there is also the rule $R_{\Gamma[i]}$: "If $\Gamma[i]$ is at any node, add $-n_i - m_i$ to $H$." There are also soft rules $R_a$, $R_A$, and $R_{root}$, defined as in the context-free case.

## Acknowledgements

I am grateful to Géraldine Legendre, Yoshiro Miyata, and Alan Prince for many helpful discussions. The research presented here has been supported in part by NSF grant BS-9209265 and by the University of Colorado at Boulder Council on Research and Creative Work.

# References

Cohen, M. A. and Grossberg, S. (1983). Absolute stability of global pattern formation and parallel memory storage by competitive neural networks. *IEEE Transactions on Systems, Man, and Cybernetics*, 13:815–825.

Dolan, C. P. and Smolensky, P. (1989). Tensor Product Production System: A modular architecture and representation. *Connection Science*, 1:53–68.

Gazdar, G., Klein, E., Pullum, G., and Sag, I. (1985). *Generalized Phrase Structure Grammar*. Harvard University Press, Cambridge, MA.

Golden, R. M. (1986). The "Brain-State-in-a-Box" neural model is a gradient descent algorithm. *Mathematical Psychology*, 30–31:73–80.

Golden, R. M. (1988). A unified framework for connectionist systems. *Biological Cybernetics*, 59:109–120.

Goldsmith, J. A. (1990). *Autosegmental and Metrical Phonology*. Basil Blackwell, Oxford.

Goldsmith, J. A. (In press). Phonology as an intelligent system. In Napoli, D. J. and Kegl, J. A., editors, *Bridges between Psychology and Linguistics: A Swarthmore Festschrift for Lila Gleitman*. Cambridge University Press, Cambridge.

Hinton, G. E. and Sejnowski, T. J. (1983). Analyzing cooperative computation. In *Proceedings of the Fifth Annual Conference of the Cognitive Science Society*, Rochester, NY. Erlbaum Associates.

Hinton, G. E. and Sejnowski, T. J. (1986). Learning and relearning in Boltzmann machines. In Rumelhart, D. E., McClelland, J. L., and the PDP Research Group, editors, *Parallel Distributed Processing: Explorations in the Microstructure of Cognition, Volume 1: Foundations*, chapter 7, pages 282–317. MIT Press/Bradford Books, Cambridge, MA.

Hopcroft, J. E. and Ullman, J. D. (1979). *Introduction to Automata Theory, Languages, and Computation*. Addison-Wesley, Reading, MA.

Hopfield, J. J. (1982). Neural networks and physical systems with emergent collective computational abilities. *Proceedings of the National Academy of Sciences, USA*, 79:2554–2558.

Hopfield, J. J. (1984). Neurons with graded response have collective computational properties like those of two-state neurons. *Proceedings of the National Academy of Sciences, USA*, 81:3088–3092.

Hopfield, J. J. (1987). Learning algorithms and probability distributions in feedforward and feed-back networks. *Proceedings of the National Academy of Sciences, USA*, 84:8429–8433.

Lakoff, G. (1988). A suggestion for a linguistics with connectionist foundations. In Touretzky, D., Hinton, G. E., and Sejnowski, T. J., editors, *Proceedings of the Connectionist Models Summer School*, pages 301–314, San Mateo, CA. Morgan Kaufmann.

Lakoff, G. (1989). Cognitive phonology. Paper presented at the UC-Berkeley Workshop on Rules and Constraints.

Legendre, G., Miyata, Y., and Smolensky, P. (1990a). Harmonic Grammar—A formal multi-level connectionist theory of linguistic well-formedness: Theoretical foundations. In *Proceedings of the Twelfth Annual Conference of the Cognitive Science Society*, pages 388–395, Cambridge, MA. Lawrence Erlbaum.

Legendre, G., Miyata, Y., and Smolensky, P. (1990b). Harmonic Grammar—A formal multi-level connectionist theory of linguistic well-formedness: An application. In *Proceedings of the Twelfth Annual Conference of the Cognitive Science Society*, pages 884–891, Cambridge, MA. Lawrence Erlbaum.

Legendre, G., Miyata, Y., and Smolensky, P. (1991a). Distributed recursive structure processing. In Touretzky, D. S. and Lippman, R., editors, *Advances in Neural Information Processing Systems 3*, pages 591–597, San Mateo, CA. Morgan Kaufmann. Slightly expanded version in Brian Mayoh, editor, *Scandinavian Conference on Artificial Intelligence—91*, pages 47–53. IOS Press, Amsterdam.

Legendre, G., Miyata, Y., and Smolensky, P. (1991b). Unifying syntactic and semantic approaches to unaccusativity: A connectionist approach. In Sutton, L. and Johnson (with Ruth Shields), C., editors, *Proceedings of the Seventeenth Annual Meeting of the Berkeley Linguistics Society*, pages 156–167, Berkeley, CA.

Legendre, G., Miyata, Y., and Smolensky, P. (In press). Can connectionism contribute to syntax? Harmonic Grammar, with an application. In Deaton, K., Noske, M., and Ziolkowski, M., editors, *Proceedings of the 26th Meeting of the Chicago Linguistic Society*, Chicago, IL.

Prince, A. and Smolensky, P. (1991). Notes on connectionism and Harmony Theory in linguistics. Technical report, Department of Computer Science, University of Colorado at Boulder. Technical Report CU-CS-533-91.

Prince, A. and Smolensky, P. (In preparation). Optimality Theory: Constraint interaction in generative grammar.

Smolensky, P. (1983). Schema selection and stochastic inference in modular environments. In *Proceedings of the National Conference on Artificial Intelligence*, pages 378–382, Washington, DC.

Smolensky, P. (1986). Information processing in dynamical systems: Foundations of Harmony Theory. In Rumelhart, D. E., McClelland, J. L., and the PDP Research Group, editors, *Parallel Distributed Processing: Explorations in the Microstructure of Cognition. Volume 1: Foundations*, chapter 6, pages 194–281. MIT Press/Bradford Books, Cambridge, MA.

Smolensky, P. (1987). On variable binding and the representation of symbolic structures in connectionist systems. Technical report, Department of Computer Science, University of Colorado at Boulder. Technical Report CU-CS-355-87.

Smolensky, P. (1990). Tensor product variable binding and the representation of symbolic structures in connectionist networks. *Artificial Intelligence*, 46:159–216.

Smolensky, P., Legendre, G., and Miyata, Y. (1992). Principles for an integrated connectionist/symbolic theory of higher cognition. Technical report, Department of Computer Science, University of Colorado at Boulder. Technical Report CU-CS-600-92.
